# Improved Gaussian Mixture Density Estimates Using Bayesian Penalty Terms and Network Averaging

**Dirk Ormoneit**
Institut für Informatik (H2)
Technische Universität München
80290 München, Germany
*ormoneit@informatik.tu-muenchen.de*

**Volker Tresp**
Siemens AG
Central Research
81730 München, Germany
*Volker.Tresp@zfe.siemens.de*

## Abstract

We compare two regularization methods which can be used to improve the generalization capabilities of Gaussian mixture density estimates. The first method uses a Bayesian prior on the parameter space. We derive EM (Expectation Maximization) update rules which maximize the a posterior parameter probability. In the second approach we apply ensemble averaging to density estimation. This includes Breiman's "bagging", which recently has been found to produce impressive results for classification networks.

## 1 Introduction

Gaussian mixture models have recently attracted wide attention in the neural network community. Important examples of their application include the training of radial basis function classifiers, learning from patterns with missing features, and active learning. The appeal of Gaussian mixtures is based to a high degree on the applicability of the EM (Expectation Maximization) learning algorithm, which may be implemented as a fast neural network learning rule ([Now91], [Orm93]). Severe problems arise, however, due to singularities and local maxima in the log-likelihood function. Particularly in high-dimensional spaces these problems frequently cause the computed density estimates to possess only relatively limited generalization capabilities in terms of predicting the densities of new data points. As shown in this paper, considerably better generalization can be achieved using regularization.

We will compare two regularization methods. The first one uses a Bayesian prior on the parameters. By using conjugate priors we can derive EM learning rules for finding the MAP (maximum a posteriori probability) parameter estimate. The second approach consists of averaging the outputs of ensembles of Gaussian mixture density estimators trained on identical or resampled data sets. The latter is a form of "bagging" which was introduced by Breiman ([Bre94]) and which has recently been found to produce impressive results for classification networks. By using the regularized density estimators in a Bayes classifier ([THA93], [HT94], [KL95]), we demonstrate that both methods lead to density estimates which are superior to the unregularized Gaussian mixture estimate.

## 2   Gaussian Mixtures and the EM Algorithm

Consider the problem of estimating the probability density of a continuous random vector $x \in \mathcal{R}^d$ based on a set $x^* = \{x^k | 1 \leq k \leq m\}$ of iid. realizations of $x$. As a density model we choose the class of Gaussian mixtures $p(x|\Theta) = \sum_{i=1}^{n} \kappa_i p(x|i, \mu_i, \Sigma_i)$, where the restrictions $\kappa_i \geq 0$ *and* $\sum_{i=1}^{n} \kappa_i = 1$ apply. $\Theta$ denotes the parameter vector $(\kappa_i, \mu_i, \Sigma_i)_{i=1}^{n}$. The $p(x|i, \mu_i, \Sigma_i)$ are multivariate normal densities:

$$p(x|i, \mu_i, \Sigma_i) = (2\pi)^{-\frac{d}{2}} |\Sigma_i|^{-1/2} \exp\left[-1/2(x - \mu_i)^t \Sigma_i^{-1}(x - \mu_i)\right].$$

The Gaussian mixture model is well suited to approximate a wide class of continuous probability densities. Based on the model and given the data $x^*$, we may formulate the log-likelihood as

$$l(\Theta) = \log\left[\prod_{k=1}^{m} p(x^k|\Theta)\right] = \sum_{k=1}^{m} \log \sum_{i=1}^{n} \kappa_i p(x^k|i, \mu_i, \Sigma_i).$$

Maximum likelihood parameter estimates $\hat{\Theta}$ may efficiently be computed with the EM (Expectation Maximization) algorithm ([DLR77]). It consists of the iterative application of the following two steps:

1. In the E-step, based on the current parameter estimates, the posterior probability that unit $i$ is responsible for the generation of pattern $x^k$ is estimated as

$$h_i^k = \frac{\kappa_i p(x^k|i, \mu_i, \Sigma_i)}{\sum_{i'=1}^{n} \kappa_{i'} p(x^k|i', \mu_{i'}, \Sigma_{i'})}. \tag{1}$$

2. In the M-step, we obtain new parameter estimates (denoted by the prime):

$$\kappa_i' = \frac{1}{m} \sum_{k=1}^{m} h_i^k \quad (2) \qquad \mu_i' = \frac{\sum_{k=1}^{m} h_i^k x^k}{\sum_{l=1}^{m} h_i^l} \quad (3)$$

$$\Sigma_i' = \frac{\sum_{k=1}^{m} h_i^k (x^k - \mu_i')(x^k - \mu_i')^t}{\sum_{l=1}^{m} h_i^l}. \tag{4}$$

Note that $\kappa_i'$ is a scalar, whereas $\mu_i'$ denotes a $d$-dimensional vector and $\Sigma_i'$ is a $d \times d$ matrix.

It is well known that training neural networks as predictors using the maximum likelihood parameter estimate leads to overfitting. The problem of overfitting is even more severe in density estimation due to singularities in the log-likelihood function. Obviously, the model likelihood becomes infinite in a trivial way if we concentrate all the probability mass on one or several samples of the training set.

In a Gaussian mixture this is just the case if the center of a unit coincides with one of the data points and $\Sigma$ approaches the zero matrix. Figure 1 compares the true and the estimated probability density in a toy problem. As may be seen, the contraction of the Gaussians results in (possibly infinitely) high peaks in the Gaussian mixture density estimate. A simple way to achieve numerical stability is to artificially enforce a lower bound on the diagonal elements of $\Sigma$. This is a very rude way of regularization, however, and usually results in low generalization capabilities. The problem becomes even more severe in high-dimensional spaces. To yield reasonable approximations, we will apply two methods of regularization, which will be discussed in the following two sections.

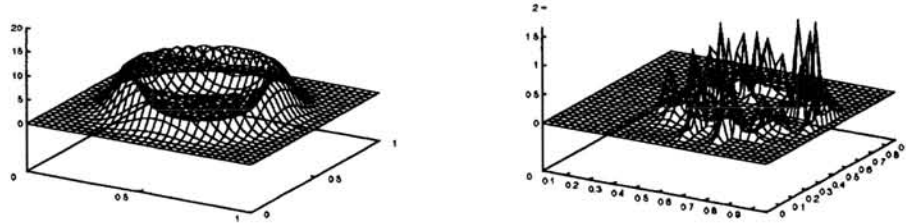

Figure 1: *True density (left) and unregularized density estimation (right).*

## 3  Bayesian Regularization

In this section we propose a Bayesian prior distribution on the Gaussian mixture parameters, which leads to a numerically stable version of the EM algorithm. We first select a family of prior distributions on the parameters which is conjugate*. Selecting a conjugate prior has a number of advantages. In particular, we obtain analytic solutions for the posterior density and the predictive density. In our case, the posterior density is a complex mixture of densities[†]. It is possible, however, to derive EM-update rules to obtain the MAP parameter estimates.

A conjugate prior of a single multivariate normal density is a product of a normal density $N(\mu_i|\hat{\mu}, \eta^{-1}\Sigma_i)$ and a Wishart density $Wi(\Sigma_i^{-1}|\alpha, \beta)$ ([Bun94]). A proper conjugate prior for the the mixture weightings $\kappa = (\kappa_1, ..., \kappa_n)$ is a Dirichlet density $D(\kappa|\gamma)$[‡]. Consequently, the prior of the overall Gaussian mixture is the product $D(\kappa|\gamma)\prod_{i=1}^{n} N(\mu_i|\hat{\mu}, \eta^{-1}\Sigma_i)Wi(\Sigma_i^{-1}|\alpha, \beta)$. Our goal is to find the MAP parameter estimate, that is parameters which assume the maximum of the log-posterior

$$
\begin{aligned}
l_p(\Theta) &= \sum_{k=1}^{m} \log \sum_{i=1}^{n} \kappa_i p(x^k|i, \mu_i, \Sigma_i) + \log D(\kappa|\gamma) \\
&\quad + \sum_{i=1}^{n} [\log N(\mu_i|\hat{\mu}, \eta^{-1}\Sigma_i) + \log Wi(\Sigma_i^{-1}|\alpha, \beta)].
\end{aligned}
$$

As in the unregularized case, we may use the EM-algorithm to find a local maximum

---

*A family $F$ of probability distributions on $\Theta$ is said to be *conjugate* if, for every $\pi \in F$, the posterior $\pi(\Theta|x)$ also belongs to $F$ ([Rob94]).

[†]The posterior distribution can be written as a sum of $n^m$ simple terms.

[‡]Those densities are defined as follows ($b$ and $c$ are normalizing constants):

$$
D(\kappa|\gamma) = b\prod_{i=1}^{n} \kappa_i^{\gamma_i - 1}, \text{ with } \kappa_i \geq 0 \text{ and } \sum_{i=1}^{n} \kappa_i = 1
$$

$$
N(\mu_i|\hat{\mu}, \eta^{-1}\Sigma_i) = (2\pi)^{-\frac{d}{2}}|\eta^{-1}\Sigma_i|^{-1/2} \exp\left[-\frac{\eta}{2}(\mu_i - \hat{\mu})^t\Sigma_i^{-1}(\mu_i - \hat{\mu})\right]
$$

$$
Wi(\Sigma_i^{-1}|\alpha, \beta) = c|\Sigma_i^{-1}|^{\alpha-(d+1)/2} \exp\left[-tr(\beta\Sigma_i^{-1})\right].
$$

of $l_p(\Theta)$. The E-step is identical to (1). The M-step becomes

$$\kappa_i' = \frac{\sum_{k=1}^m h_i^k + \gamma_i - 1}{m + \sum_{i=1}^n \gamma_i - n} \qquad (5) \qquad \mu_i' = \frac{\sum_{k=1}^m h_i^k x^k + \eta\hat{\mu}}{\sum_{l=1}^m h_i^l + \eta} \qquad (6)$$

$$\Sigma_i' = \frac{\sum_{k=1}^m h_i^k (x^k - \mu_i')(x^k - \mu_i')^t + \eta(\mu_i' - \hat{\mu})(\mu_i' - \hat{\mu})^t + 2\beta}{\sum_{l=1}^m h_i^l + 2\alpha - d}. \qquad (7)$$

As typical for conjugate priors, prior knowledge corresponds to a set of artificial training data which is also reflected in the EM-update equations. In our experiments, we focus on a prior on the variances which is implemented by $\beta \neq 0$, where $0$ denotes the $d \times d$ zero matrix. All other parameters we set to "neutral" values:

$$\gamma_i = 1 \ \forall i : 1 \leq i \leq n, \quad \alpha = (d+1)/2, \quad \eta = 0, \quad \beta = \bar{\beta}I^d$$

$I^d$ is the $d \times d$ unity matrix. The choice of $\alpha$ introduces a bias which favors large variances[§]. The effect of various values of the scalar $\bar{\beta}$ on the density estimate is illustrated in figure 2. Note that if $\bar{\beta}$ is chosen too small, overfitting still occurs. If it is chosen to large, on the other hand, the model is too constraint to recognize the underlying structure.

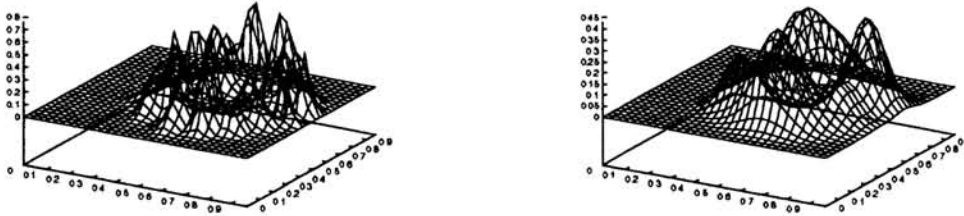

Figure 2: *Regularized density estimates (left:* $\bar{\beta} = 0.05$*, right:* $\bar{\beta} = 0.1$*).*

Typically, the optimal value for $\bar{\beta}$ is not known a priori. The simplest procedure consists of using that $\bar{\beta}$ which leads to the best performance on a validation set, analogous to the determination of the optimal weight decay parameter in neural network training. Alternatively, $\bar{\beta}$ might be determined according to appropriate Bayesian methods ([Mac91]). Either way, only few additional computations are required for this method if compared with standard EM.

## 4   Averaging Gaussian Mixtures

In this section we discuss the averaging of several Gaussian mixtures to yield improved probability density estimation. The averaging over neural network ensembles has been applied previously to regression and classification tasks ([PC93]).

There are several different variants on the simple averaging idea. First, one may train all networks on the complete set of training data. The only source of disagreement between the individual predictions consists in different local solutions found by the likelihood maximization procedure due to different starting points. Disagreement is essential to yield an improvement by averaging, however, so that this proceeding only seems advantageous in cases where the relation between training data and weights is extremely non-deterministic in the sense that in training,

---

[§]If $\lambda$ is distributed according to $Wi(\lambda|\alpha, \beta)$, then $E[\lambda^{-1}] = (\alpha - (d+1)/2)^{-1}\beta$. In our case $\lambda$ is $\Sigma_i^{-1}$, so that $E[\Sigma_i] \to \infty \cdot \beta$ for $\alpha \to (d+1)/2$.

different solutions are found from different random starting points. A straightforward way to increase the disagreement is to train each network on a resampled version of the original data set. If we resample the data *without replacement*, the size of each training set is reduced, in our experiments to 70% of the original. The averaging of neural network predictions based on resampling *with replacement* has recently been proposed under the notation "bagging" by Breiman ([Bre94]), who has achieved dramatically improved results in several classification tasks. He also notes, however, that an actual improvement of the prediction can only result if the estimation procedure is relatively unstable. As discussed, this is particularly the case for Gaussian mixture training. We therefore expect bagging to be well suited for our task.

## 5 Experiments and Results

To assess the practical advantage resulting from regularization, we used the density estimates to construct classifiers and compared the resulting prediction accuracies using a toy problem and a real-world problem. The reason is that the generalization error of density estimates in terms of the likelihood based on the test data is rather unintuitive whereas performance on a classification problem provides a good impression of the degree of improvement. Assume we have a set of N labeled data $z^* = \{(x^k, l^k)|k = 1, ..., N\}$, where $l_k \in \Upsilon = \{1, ..., C\}$ denotes the class label of each input $x^k$. A classifier of new inputs $x$ is yielded by choosing the class $l$ with the maximum posterior class-probability $p(l|x)$. The posterior probabilities may be derived from the class-conditional data likelihood $p(x|l)$ via Bayes theorem: $p(l|x) = p(x|l)p(l)/p(x) \propto p(x|l)p(l)$. The resulting partitions of the input space are optimal for the true $p(l|x)$. A viable way to approximate the posterior $p(l|x)$ is to estimate $p(x|l)$ and $p(l)$ from the sample data.

### 5.1 Toy Problem

In the toy classification problem the task is to discriminate the two classes of circulatory arranged data shown in figure 3. We generated 200 data points for each class and subdivided them into two sets of 100 data points. The first was used for training, the second to test the generalization performance. As a network architecture we chose a Gaussian mixture with 20 units. Table 1 summarizes the results, beginning with the unregularized Gaussian mixture which is followed by the averaging and the Bayesian penalty approaches. The three rows for averaging correspond to the results yielded without applying resampling (local max.), with resampling with-

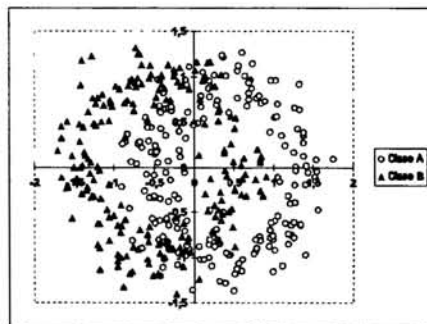

Figure 3: *Toy Classification Task.*

out replacement (70% subsets), and with resampling with replacement (bagging). The performances on training and test set are measured in terms of the model log-likelihood. Larger values indicate a better performance. We report separate results for class A and B, since the densities of both were estimated separately. The final column shows the prediction accuracy in terms of the percentage of correctly classified data in the test set. We report the average results from 20 experiments. The numbers in brackets denote the standard deviations $\sigma$ of the results. Multiplying $\sigma$ with $\tau_{19;95\%}/\sqrt{20} = 0.4680$ yields 95% confidence intervals. The best result in each category is underlined.

| Algorithm | Log-Likelihood | | | | Accuracy |
| | Training | | Test | | |
| | A | B | A | B | |
|---|---|---|---|---|---|
| unreg. | -120.8 (13.3) | -120.4 (10.8) | -224.9 (32.6) | -241.9 (34.1) | 80.6% (2.8) |
| Averaging: | | | | | |
| local max. | -115.6 (6.0) | -112.6 (6.6) | -200.9 (13.9) | -209.1 (16.3) | 81.8% (3.1) |
| 70% subset | -106.8 (5.8) | -105.1 (6.7) | -188.8 (9.5) | -196.4 (11.3) | 83.2% (2.9) |
| bagging | -83.8 (4.9) | -83.1 (7.1) | -194.2 (7.3) | -200.1 (11.3) | 82.6% (3.4) |
| Penalty: | | | | | |
| $\bar{\beta} = 0.01$ | -149.3 (18.5) | -146.5 (5.9) | -186.2 (13.9) | -182.9 (11.6) | 83.1% (2.9) |
| $\bar{\beta} = 0.02$ | -156.0 (16.5) | -153.0 (4.8) | -177.1 (11.8) | -174.9 (7.0) | 84.4% (6.3) |
| $\bar{\beta} = 0.05$ | -173.9 (24.3) | -167.0 (15.8) | -182.0 (20.1) | -173.9 (14.3) | 81.5% (5.9) |
| $\bar{\beta} = 0.1$ | -183.0 (21.9) | -181.9 (21.1) | -184.6 (21.0) | -182.5 (21.1) | 78.5% (5.1) |

Table 1: Performances in the toy classification problem .

As expected, all regularization methods outperform the maximum likelihood approach in terms of correct classification. The performance of the Bayesian regularization is hereby very sensitive to the appropriate choice of the regularization parameter $\bar{\beta}$. Optimality of $\bar{\beta}$ with respect to the density prediction and optimality with respect to prediction accuracy on the test set roughly coincide (for $\bar{\beta} = 0.02$). Averaging is inferior to the Bayesian approach if an optimal $\bar{\beta}$ is chosen.

## 5.2   BUPA Liver Disorder Classification

As a second task we applied our methods to a real-world decision problem from the medical environment. The problem is to detect liver disorders which might arise from excessive alcohol consumption. Available information consists of five blood tests as well as a measure of the patients' daily alcohol consumption. We subdivided the 345 available samples into a training set of 200 and a test set of 145 samples. Due to the relatively few data we did not try to determine the optimal regularization parameter using a validation process and will report results on the test set for different parameter values.

| Algorithm | Accuracy |
|---|---|
| unregularized | 64.8 % |
| Bayesian penalty ($\beta = 0.05$) | 65.5 % |
| Bayesian penalty ($\bar{\beta} = 0.10$) | 66.9 % |
| Bayesian penalty ($\bar{\beta} = 0.20$) | 61.4 % |
| averaging (local maxima) | 65.5 % |
| averaging (70 % subset) | 72.4 % |
| averaging (bagging) | 71.0 % |

Table 2: Performances in the liver disorder classification problem .

The results of our experiments are shown in table 2. Again, both regularization methods led to an improvement in prediction accuracy. In contrast to the toy problem, the averaged predictor was superior to the Bayesian approach here. Note that the resampling led to an improvement of more than five percent points compared to unresampled averaging.

# 6   Conclusion

We proposed a Bayesian and an averaging approach to regularize Gaussian mixture density estimates. In comparison with the maximum likelihood solution both approaches led to considerably improved results as demonstrated using a toy problem and a real-world classification task. Interestingly, none of the methods outperformed the other in both tasks. This might be explained with the fact that Gaussian mixture density estimates are particularly unstable in high-dimensional spaces with relatively few data. The benefit of averaging might thus be greater in this case. Averaging proved to be particularly effective if applied in connection with resampling of the training data, which agrees with results in regression and classification tasks. If compared to Bayesian regularization, averaging is computationally expensive. On the other hand, Baysian approaches typically require the determination of hyper parameters (in our case $\bar{\beta}$), which is not the case for averaging approaches.

# References

[Bre94]   L. Breiman. Bagging predictors. Technical report, UC Berkeley, 1994.

[Bun94]   W. Buntine. Operations for learning with graphical models. *Journal of Artificial Intelligence Research*, 2:159–225, 1994.

[DLR77]   A. P. Dempster, N. M. Laird, and D. B. Rubin. Maximum likelihood from incomplete data via the EM algorithm. *J. Royal Statistical Society B*, 1977.

[HT94]    T. Hastie and R. Tibshirani. Discriminant analysis by gaussian mixtures. Technical report, AT&T Bell Labs and University of Toronto, 1994.

[KL95]    N. Kambhatla and T. K. Leen. Classifying with gaussian mixtures and clusters. In *Advances in Neural Information Processing Systems 7*. Morgan Kaufman, 1995.

[Mac91]   D. MacKay. *Bayesian Modelling and Neural Networks*. PhD thesis, California Institute of Technology, Pasadena, 1991.

[Now91]   S. J. Nowlan. *Soft Competitive Adaption: Neural Network Learning Algorithms based on Fitting Statistical Mixtures*. PhD thesis, School of Computer Science, Carnegie Mellon University, Pittsburgh, 1991.

[Orm93]   D. Ormoneit. Estimation of probability densities using neural networks. Master's thesis, Technische Universität München, 1993.

[PC93]    M. P. Perrone and L. N. Cooper. When networks disagree: Ensemble methods for hybrid Neural networks. In *Neural Networks for Speech and Image Processing*. Chapman Hall, 1993.

[Rob94]   C. P. Robert. *The Bayesian Choice*. Springer-Verlag, 1994.

[THA93]   V. Tresp, J. Hollatz, and S. Ahmad. Network structuring and training using rule-based knowledge. In *Advances in Neural Information Processing Systems 5*. Morgan Kaufman, 1993.
